# Estimation of Information Theoretic Measures for Continuous Random Variables

**Fernando Pérez-Cruz**
Princeton University, Electrical Engineering Department
B-311 Engineering Quadrangle, 08544 Princeton (NJ)
fp@princeton.edu

## Abstract

We analyze the estimation of information theoretic measures of continuous random variables such as: differential entropy, mutual information or Kullback-Leibler divergence. The objective of this paper is two-fold. First, we prove that the information theoretic measure estimates using the $k$-nearest-neighbor density estimation with fixed $k$ converge almost surely, even though the $k$-nearest-neighbor density estimation with fixed $k$ does not converge to its true measure. Second, we show that the information theoretic measure estimates do not converge for $k$ growing linearly with the number of samples. Nevertheless, these nonconvergent estimates can be used for solving the two-sample problem and assessing if two random variables are independent. We show that the two-sample and independence tests based on these nonconvergent estimates compare favorably with the maximum mean discrepancy test and the Hilbert Schmidt independence criterion.

## 1 Introduction

Kullback-Leibler divergence, mutual information and differential entropy are central to information theory [5]. The divergence [17] measures the 'distance' between two density distributions while mutual information measures the information one random variable contains about a related random variable [23]. In machine learning, statistics and neuroscience the information theoretic measures also play a leading role. For instance, the divergence is the error exponent in large deviation theory [5] and the divergence can be directly applied to solving the two-sample problem [1]. Mutual information is extensively used to assess whether two random variables are independent [2] and has been proposed to solve the all-relevant feature selection problem [8, 24]. Information-theoretic analysis of neural data is unavoidable given the questions neurophysiologists are interested in[1]. There are other relevant applications in different research areas in which divergence estimation is used to measure the difference between two density functions, such as multimedia [19] and text [13] classification, among others.

The estimation of information theoretic quantities can be traced back to the late fifties [7], when Dobrushin estimated the differential entropy for one-dimensional random variables. The review paper by Beirlant et al. [4] analyzes the different contributions of nonparametric differential entropy estimation for continuous random variables. The estimation of the divergence and mutual information for continuous random variables has been addressed by many different authors [25, 6, 26, 18, 20, 16], see also the references therein. Most of these approaches are based on estimating the densities first. For example, in [25], the authors propose to estimate the densities based on data-dependent histograms with a fixed number of samples from $q(\mathbf{x})$ in each bin. The authors of [6] compute relative frequencies on data-driven partitions achieving local independence for estimating mutual information. Also, in [20, 21], the authors compute the divergence using a variational approach, in which

convergence is proven ensuring that the estimate for $p(\mathbf{x})/q(\mathbf{x})$ or $\log p(\mathbf{x})/q(\mathbf{x})$ converges to the true measure ratio or its $\log$ ratio.

There are only a handful of approaches that use $k$-nearest-neighbors ($k$-nn) density estimation [26, 18, 16] for estimating the divergence and mutual information for finite $k$. Although finite $k$-nn density estimation does not converge to the true measure, the authors are able to prove mean-square consistency of their divergence estimators imposing some regularity constraint over the densities. These proofs are based on the results reported in [15] for estimating the differential entropy with $k$-nn density estimation.

The results in this paper are two-fold. First, we prove almost sure convergence of our divergence estimate based on $k$-nn density estimation with finite $k$. Our result is based on describing the statistics of $p(\mathbf{x})/\widehat{p}(\mathbf{x})$ as a waiting time distribution independent of $p(\mathbf{x})$. We can readily apply this result to the estimation of the differential entropy and mutual information.

Second, we show that for $k$ linearly growing with the number of samples, our estimates do not converge nor present known statistics. But they can be reliably used for solving the two-sample problem or assessing if two random variables are independent. We show that for this choice of $k$, the estimates of the divergence or mutual information perform, respectively, as well as the maximum mean discrepancy (MMD) test in [9] and the Hilbert Schmidt independence criterion (HSIC) proposed in [10].

The rest of the paper is organized as follows. We prove in Section 2 the almost sure convergence of the divergence estimate based on $k$-nn density estimation with fixed $k$. We extend this result for differential entropy and mutual information in Section 3. In Section 4 we present some examples to illustrate the convergence of our estimates and to show how can they be used to assess the independence of related random variables. Section 5 concludes the paper with some final remarks.

## 2  Estimation of the Kullback-Leibler Divergence

If the densities $P$ and $Q$ exist with respect to a Lebesgue measure, the Kullback-Leibler divergence is given by:

$$D(P||Q) = \int_{\mathbb{R}^d} p(\mathbf{x}) \log \frac{p(\mathbf{x})}{q(\mathbf{x})} d\mathbf{x} \geq 0. \tag{1}$$

This divergence is finite whenever $P$ is absolutely continuous with respect to $Q$ and it is zero only if $P = Q$.

The idea of using $k$-nn density estimation to estimate the divergence was put forward in [26, 18], where they prove mean-square consistency of their estimator for finite $k$. In this paper, we prove the almost sure convergence of this divergence estimator, using waiting-times distributions without needing to impose additional conditions over the density models. Given a set with $n$ i.i.d. samples from $p(\mathbf{x})$, $\mathcal{X} = \{\mathbf{x}_i\}_{i=1}^n$, and $m$ i.i.d. samples from $q(\mathbf{x})$, $\mathcal{X}' = \{\mathbf{x}'_j\}_{j=1}^m$, we estimate $D(P||Q)$ from a $k$-nn density estimate of $p(\mathbf{x})$ and $q(\mathbf{x})$ as follows:

$$\widehat{D}_k(P||Q) = \frac{1}{n} \sum_{i=1}^n \log \frac{\widehat{p}_k(\mathbf{x}_i)}{\widehat{q}_k(\mathbf{x}_i)} = \frac{d}{n} \sum_{i=1}^n \log \frac{s_k(\mathbf{x}_i)}{k_k(\mathbf{x}_i)} + \log \frac{m}{n-1} \tag{2}$$

where

$$\widehat{p}_k(\mathbf{x}_i) = \frac{k}{(n-1)} \frac{\Gamma(d/2+1)}{\pi^{d/2}} \frac{1}{r_k(\mathbf{x}_i)^d} \tag{3}$$

$$\widehat{q}_k(\mathbf{x}_i) = \frac{k}{m} \frac{\Gamma(d/2+1)}{\pi^{d/2}} \frac{1}{s_k(\mathbf{x}_i)^d} \tag{4}$$

$r_k(\mathbf{x}_i)$ and $s_k(\mathbf{x}_i)$ are, respectively, the Euclidean distances to the $k$-nn of $\mathbf{x}_i$ in $\mathcal{X} \backslash \mathbf{x}_i$ and $\mathcal{X}'$, and $\pi^{d/2}/\Gamma(d/2+1)$ is the volume of the unit-ball in $\mathbb{R}^d$. Before proving (2) converges almost surely to $D(P||Q)$, let us show an intermediate necessary result.

**Lemma 1.** *Given $n$ i.i.d. samples, $\mathcal{X} = \{\mathbf{x}_i\}_{i=1}^n$, from an absolutely continuous probability distribution $P$, the limiting distribution of $p(\mathbf{x})/\widehat{p}_1(\mathbf{x})$ is exponentially distributed with unit mean for any $\mathbf{x}$ in the support of $p(\mathbf{x})$.*

*Proof.* Let's initially assume $p(\mathbf{x})$ is a $d$-dimensional uniform distribution with a given support. The set $\mathcal{S}_{\mathbf{x},R} = \{\mathbf{x}_i | \, \|\mathbf{x}_i - \mathbf{x}\|_2 \leq R, \mathbf{x}_i \in \mathcal{X}\}$ contains all the samples from $\mathcal{X}$ inside the ball centered in $\mathbf{x}$ of radius $R$. The radius $R$ has to be small enough for the ball centered in $\mathbf{x}$ to be contained within the support of $p(\mathbf{x})$.

The samples in $\{\|\mathbf{x}_i - \mathbf{x}\|_2^d | \, \mathbf{x}_i \in \mathcal{S}_{\mathbf{x},R}\}$ are consequently uniformly distributed between 0 and $R^d$. Thereby, the limiting distribution of $r_1(\mathbf{x})^d = \min_{\mathbf{x}_j \in \mathcal{S}_{\mathbf{x},R}}(\|\mathbf{x}_j - \mathbf{x}\|_2^d)$ is exponentially distributed, as it measures the waiting time between the origin and the first event of a uniformly-spaced sample (see Theorem 2.4 in [3]). Since $p(\mathbf{x})n\pi^{d/2}/\Gamma(d/2 + 1)$ is the mean number of samples per unit ball centered in $\mathbf{x}$, $p(\mathbf{x})/\widehat{p}_1(\mathbf{x})$ is distributed as a unit-mean exponential distribution as $n$ tends to infinity.

For non-uniform absolutely-continuous $P$, $\mathbb{P}(r_1(\mathbf{x}) > \varepsilon) \to 0$ as $n \to \infty$ for any $\mathbf{x}$ in the support of $p(\mathbf{x})$ and any $\varepsilon > 0$. Therefore, as $n$ tends to infinity $p(\arg\min_{\mathbf{x}_j \in \mathcal{S}_{\mathbf{x},R}}(\|\mathbf{x}_j - \mathbf{x}\|_2^d)) \to p(\mathbf{x})$ and the limiting distribution of $p(\mathbf{x})/\widehat{p}_1(\mathbf{x})$ is a unit-mean exponential distribution. $\qquad\square$

**Corolary 1.** *Given $n$ i.i.d. samples, $\mathcal{X} = \{\mathbf{x}_i\}_{i=1}^n$, from an absolutely continuous probability distribution $P$, the limiting distribution of $p(\mathbf{x})/\widehat{p}_k(\mathbf{x})$ is a unit-mean $1/k$-variance gamma distribution for any $\mathbf{x}$ in the support of $p(\mathbf{x})$.*

*Proof.* In the previous proof, instead of measuring the waiting time to the first event, we compute the waiting time to the $k^{th}$ event of a uniformly-spaced sample. This waiting-time limiting distribution is a unit-mean and $1/k$-variance Erlang (gamma) distribution [14]. $\qquad\square$

**Corolary 2.** *Given $n$ i.i.d. samples, $\mathcal{X} = \{\mathbf{x}_i\}_{i=1}^n$, from an absolutely continuous probability distribution $P$, then $\widehat{p}_k(\mathbf{x}) \xrightarrow{P} p(\mathbf{x})$ for any $\mathbf{x}$ in the support of $p(\mathbf{x})$, if $k \to \infty$ and $k/n \to 0$, as $n \to \infty$.*

*Proof.* The $k$-nn in $\mathcal{X}$ tends to $\mathbf{x}$ as $k/n \to 0$ and $n \to \infty$. Thereby the limiting distribution of $p(\mathbf{x})/\widehat{p}_k(\mathbf{x})$ is a unit-mean $1/k$-variance gamma distribution. As $k \to \infty$ the variance of the gamma distribution goes to zero and consequently $\widehat{p}_k(\mathbf{x})$ converges to $p(\mathbf{x})$. $\qquad\square$

The second corollary is the widely known result that $k$-nn density estimation converges to the true measure if $k \to \infty$ and $k/n \to 0$. We have just include it in the paper for clarity and completeness. If $k$ grows linearly with $n$, the $k$-nn sample in $\mathcal{X}$ does not converge to $\mathbf{x}$, which precludes $p(\mathbf{x})/\widehat{p}_k(\mathbf{x})$ to present known statistics. For this growth on $k$, the divergence estimate does not converge to $D(P\|Q)$.

Now we can prove the almost surely convergence to (1) of the estimate in (2) based on the $k$-nn density estimation.

**Theorem 1.** *Let $P$ and $Q$ be absolutely continuous probability measures and let $P$ be absolutely continuous with respect to $Q$. Let $\mathcal{X} = \{\mathbf{x}_i\}_{i=1}^n$ and $\mathcal{X}' = \{\mathbf{x}_i'\}_{i=1}^m$ be i.i.d. samples, respectively, from $P$ and $Q$, then*

$$\widehat{D}_k(P\|Q) \qquad \xrightarrow{a.s.} \qquad D(P\|Q) \qquad (5)$$

*Proof.* We can rearrange $\widehat{D}_k(P\|Q)$ in (2) as follows:

$$\widehat{D}_k(P\|Q) = \frac{1}{n}\sum_{i=1}^n \log\frac{\widehat{p}_k(\mathbf{x}_i)}{\widehat{q}_k(\mathbf{x}_i)} = \frac{1}{n}\sum_{i=1}^n \log\frac{p(\mathbf{x}_i)}{q(\mathbf{x}_i)} - \frac{1}{n}\sum_{i=1}^n \log\frac{p(\mathbf{x}_i)}{\widehat{p}_k(\mathbf{x}_i)} + \frac{1}{n}\sum_{i=1}^n \log\frac{q(\mathbf{x}_i)}{\widehat{q}_k(\mathbf{x}_i)} \quad (6)$$

The first term is the empirical estimate of (1) and, by the law of large numbers [11], it converges almost surely to its mean, $D(P\|Q)$.

The limiting distributions of $p(\mathbf{x}_i)/\widehat{p}_k(\mathbf{x}_i)$ and $q(\mathbf{x}_i)/\widehat{q}_k(\mathbf{x}_i)$ are unit-mean $1/k$-variance gamma distributions, independent of $i$, $p(\mathbf{x})$ and $q(\mathbf{x})$ (see Corollary 1). In the large sample limit:

$$\frac{1}{n}\sum_{i=1}^n \log\frac{p(\mathbf{x}_i)}{\widehat{p}_k(\mathbf{x}_i)} \qquad \xrightarrow{a.s.} \qquad \frac{k^k}{(k-1)!}\int_0^\infty \log(z)z^{k-1}e^{-kz}dz \qquad (7)$$

by the law of large numbers [11].

Finally, the sum of almost surely convergent terms also converges almost surely [11], which completes our proof. $\qquad\square$

The $k$-nn based divergence estimator is biased, because the convergence rate of $p(\mathbf{x}_i)/\widehat{p}_k(\mathbf{x}_i)$ and $q(\mathbf{x}_i)/\widehat{q}_k(\mathbf{x}_i)$ to the unit-mean $1/k$-variance gamma distribution depends on the density models and we should not expect them to be identical. If $p(\mathbf{x}) = q(\mathbf{x})$, the divergence is zero and our estimate is unbiased for any $k$ (even if $k/n$ does not tend to zero), since the statistics of the second and third term in (6) are identical and they cancel each other out for any $n$ (their expected mean is the same). We use the Monte Carlo based test described in [9] with our divergence estimator to solve the two-sample problem and decide if the samples from $\mathcal{X}$ and $\mathcal{X}'$ actually came from the same distribution.

## 3  Differential Entropy and Mutual Information Estimation

The results obtained for the divergence can be readily applied to estimate the differential entropy of a random variable or the mutual information between two correlated random variables.

The differential entropy for an absolutely continuous random variable $P$ is given by:

$$h(\mathbf{x}) = -\int p(\mathbf{x}) \log p(\mathbf{x}) d\mathbf{x} \tag{8}$$

We can estimate the differential entropy given a set with $n$ i.i.d. samples from $P$, $\mathcal{X} = \{\mathbf{x}_i\}_{i=1}^n$, using $k$-nn density estimation as follows:

$$\widehat{h}_k(\mathbf{x}) = -\frac{1}{n} \sum_{i=1}^n \log \widehat{p}_k(\mathbf{x}_i) \tag{9}$$

where $\widehat{p}_k(\mathbf{x}_i)$ is given by (3).

**Theorem 2.** *Let $P$ be an absolutely continuous probability measure and let $\mathcal{X} = \{\mathbf{x}_i\}_{i=1}^n$ be i.i.d. samples from $P$, then*

$$\widehat{h}_k(\mathbf{x}) \xrightarrow{a.s.} h(\mathbf{x}) + \gamma_k \tag{10}$$

*where*

$$\gamma_k = -\frac{k^k}{(k-1)!} \int_0^\infty \log(z) z^{k-1} e^{-kz} dz \tag{11}$$

*and $\gamma_1 \cong 0.5772$ and it is known as the Euler-Mascheroni constant [12].*

*Proof.* We can rearrange $\widehat{h}_k(\mathbf{x})$ in (9) as follows:

$$\widehat{h}_k(\mathbf{x}) = -\frac{1}{n} \sum_{i=1}^n \log \widehat{p}_k(\mathbf{x}_i) = -\frac{1}{n} \sum_{i=1}^n \log p(\mathbf{x}_i) + \frac{1}{n} \sum_{i=1}^n \log \frac{p(\mathbf{x}_i)}{\widehat{p}_k(\mathbf{x}_i)} \tag{12}$$

The first term is the empirical estimate of (9) and, by the law of large numbers [11], it converges almost surely to its mean, $h(\mathbf{x})$.

The limiting distributions of $p(\mathbf{x}_i)/\widehat{p}_k(\mathbf{x}_i)$ is a unit-mean $1/k$-variance gamma distribution, independent of $i$ and $p(\mathbf{x})$ (see Corollary 1). In the large sample limit:

$$\frac{1}{n} \sum_{i=1}^n \log \frac{p(\mathbf{x}_i)}{\widehat{p}_k(\mathbf{x}_i)} \xrightarrow{a.s.} \frac{k^k}{(k-1)!} \int_0^\infty \log(z) z^{k-1} e^{-kz} dz = -\gamma_k \tag{13}$$

by the law of large numbers [11].

Finally, the sum of almost surely convergent terms also converges almost surely [11], which completes our proof. $\qquad\square$

Now, we can use the expansion of the conditional differential entropy, mutual information and conditional mutual information to prove the convergence of their estimates based on $k$-nn density estimation to their values.

- Conditional differential entropy:

$$h(\mathbf{y}|\mathbf{x}) = -\int p(\mathbf{x},\mathbf{y}) \log \frac{p(\mathbf{y},\mathbf{x})}{p(\mathbf{x})} d\mathbf{x} d\mathbf{y} \tag{14}$$

$$\widehat{h}(\mathbf{y}|\mathbf{x}) = -\frac{1}{n}\sum_{i=1}^{n} \log \frac{p(\mathbf{y}_i,\mathbf{x}_i)}{p(\mathbf{x}_i)} \qquad \xrightarrow{a.s.} \qquad h(\mathbf{y}|\mathbf{x}) \tag{15}$$

- Mutual Information:

$$I(\mathbf{x};\mathbf{y}) = -\int p(\mathbf{x},\mathbf{y}) \log \frac{p(\mathbf{y},\mathbf{x})}{p(\mathbf{x})p(\mathbf{y})} d\mathbf{x} d\mathbf{y} \tag{16}$$

$$\widehat{I}(\mathbf{x};|\mathbf{y}) = \frac{1}{n}\sum_{i=1}^{n} \log \frac{p(\mathbf{y}_i,\mathbf{x}_i)}{p(\mathbf{x}_i)p(\mathbf{y}_i)} \qquad \xrightarrow{a.s.} \qquad I(\mathbf{x};\mathbf{y}) + \gamma_k \tag{17}$$

- Conditional Mutual Information:

$$I(\mathbf{x};\mathbf{y}|\mathbf{z}) = \int p(\mathbf{x},\mathbf{y},\mathbf{z}) \log \frac{p(\mathbf{y},\mathbf{x},\mathbf{z})p(\mathbf{z})}{p(\mathbf{x},\mathbf{z})p(\mathbf{y},\mathbf{z})} d\mathbf{x} d\mathbf{y} d\mathbf{z} \tag{18}$$

$$\widehat{I}(\mathbf{x};\mathbf{y}|\mathbf{z}) = \frac{1}{n}\sum_{i=1}^{n} \log \frac{p(\mathbf{y}_i,\mathbf{x}_i,\mathbf{z}_i)p(\mathbf{z}_i)}{p(\mathbf{x}_i,\mathbf{z}_i)p(\mathbf{y}_i,\mathbf{z}_i)} \qquad \xrightarrow{a.s.} \qquad I(\mathbf{x};\mathbf{y}|\mathbf{z}) \tag{19}$$

## 4  Experiments

We have carried out two sets of experiments. In the first one, we show the convergence of the divergence to their limiting value as the number of samples tends to infinity and we compare the divergence estimation to the MMD test in [9] for MNIST dataset. In the second experiment, we compute if two random variables are independent and compare the obtained results to the HSIC proposed in [10].

We first compare the divergence between a uniform distribution between 0 and 1 in $d$-dimension and a zero-mean Gaussian distribution with identity covariance matrix. We plot the divergence estimates for $d = 1$ and $d = 5$ in Figure 1 as a function of $n$, for $k = 1$, $k = \sqrt{n}$ and $k = n/2$ with $m = n$.

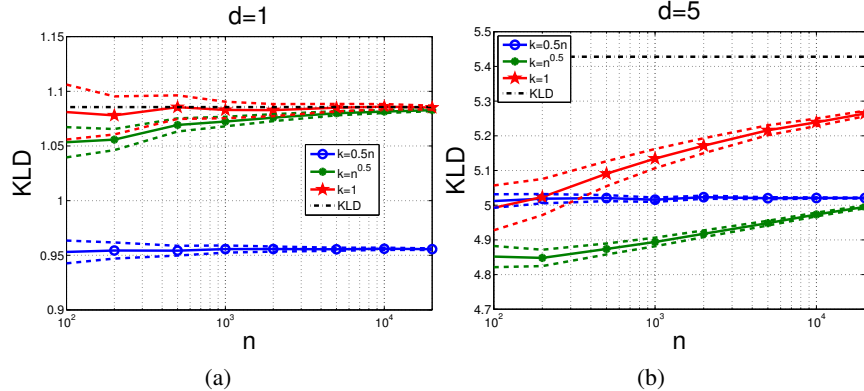

Figure 1: We plot the divergence for $d = 1$ in (a) and $d = 5$ in (b). The solid line with '$\star$' represents the divergence estimate for $k = 1$, the solid line with '$*$' represents the divergence estimate for $k = \sqrt{n}$, the solid line with '$\circ$' represents the divergence estimate for $k = n/2$ and the dashed-dotted line represents the divergence. The dashed-lines represent $\pm 3$ standard deviation for each divergence estimate. We have not added symbols to them to avoid cluttering the images further and from the plots it should be clear which confidence interval is assigned to what estimate.

As expected, the divergence estimate for $k = n/2$ does not converge to the true divergence as the limiting distributions of $p(\mathbf{x})/\widehat{p}_k(\mathbf{x})$ and $q(\mathbf{x})/\widehat{q}_k(\mathbf{x})$ are unknown and they depend on $p(\mathbf{x})$ and

$q(\mathbf{x})$, respectively. Nevertheless, this estimate converges faster to its limiting value and its variance is much smaller than that provided by the estimates of the divergence with $k = \sqrt{n}$ or $k = 1$. This may indicate that using $k = n/2$ might be a better option for solving the two-sample problem than actually trying to estimate the true divergence, as theorized in [9].

Both divergence estimates for $k = 1$ and $k = \sqrt{n}$ converge to the true divergence as the number of samples tends to infinity. The convergence of the divergence estimate for $k = 1$ is significantly faster than that with $k = \sqrt{n}$, because $p(\mathbf{x})/\widehat{p}_1(\mathbf{x})$ converges much faster to its limiting distribution than $p(\mathbf{x})/\widehat{p}_{\sqrt{n}}(\mathbf{x})$. $p(\mathbf{x})/\widehat{p}_1(\mathbf{x})$ converges faster because the nearest neighbor to $\mathbf{x}$ is much closer than the $\sqrt{n}$-nearest-neighbor and we need that the $k$-nn to be close enough to $\mathbf{x}$ for $p(\mathbf{x})/\widehat{p}_k(\mathbf{x})$ to be close to its limiting distribution. As $d$ grows the divergence estimates need many more samples to converge and even for small dimensions the number of samples can be enormously large.

Nevertheless, we can still use this divergence estimate to assess whether two sets of samples come from the same distribution, because the divergence estimate for $p(\mathbf{x}) = q(\mathbf{x})$ is unbiased for any $k$. In Figure 2(a) we plot the divergence estimate between the three's and two's handwritten digits in the MNIST dataset (http://yann.lecun.com/exdb/mnist/) in a 784 dimensional space. In Figure 2(a) we plot the divergence estimator for $\widehat{D}_1(3, 2)$ (solid line) and $\widehat{D}_1(3, 3)$ (dashed line) mean values for 100 experiments together with their 90% confidence interval. For comparison purposes we plot the MMD test from [9], in which a kernel method was proposed for solving the two-sample problem. We use the code available in http://www.kyb.mpg.de/bs/people/arthur/mmd.htm and use its bootstrap estimate for our comparisons. For $n = 5$ the error rate for the test using $k = 1$ is 1%, for $k = \sqrt{n}$ is 7% and for $k = n/2$ is 43% and for the MMD test is 34%. For $n \geq 10$ all tests reported zero error rate. It seems than the $k = 1$ test is more powerful than the MMD test in this case, at least for small $n$. But we can see that the confidence interval for the MMD test decreases faster than the test based on the divergence estimate with $k = 1$ and we should expect better performance for larger $n$, similar to the divergence estimate with $k = n/2$.

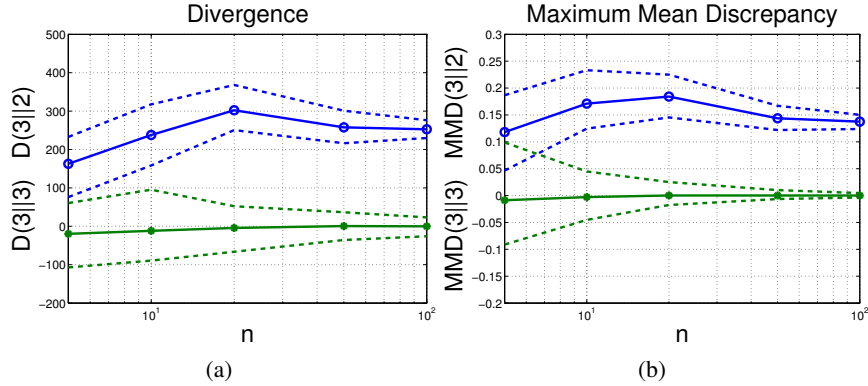

(a)                      (b)

Figure 2: In (a) we plot $\widehat{D}_1(3\|2)$ (solid), $\widehat{D}_1(3\|3)$ (dashed) and their 90% confidence interval (dotted). In (b) we repeat the same plots using the MMD test from [9].

In the second example we compute the mutual information between $y_1$ and $y_2$, which are given by:

$$\begin{bmatrix} y_1 \\ y_2 \end{bmatrix} = \begin{bmatrix} \cos(\theta) & \sin(\theta) \\ -\sin(\theta) & \cos(\theta) \end{bmatrix} \begin{bmatrix} x_1 \\ x_2 \end{bmatrix} \tag{20}$$

where $x_1$ and $x_2$ are independent and uniformly distributed between 0 and 1, and $\theta \in [0, \pi/4]$. If $\theta$ is zero, $y_1$ and $y_2$ are independent. Otherwise they are not independent, but still uncorrelated for any $\theta$.

We carry out a test for describing if $y_1$ and $y_2$ are independent. The test is identical to the one described in [10] and we use the Mote Carlo resampling technique proposed in that paper with a 95% confidence interval and 1000 repetitions. In Figure 3 we report the acceptance of the null hypothesis ($y_1$ and $y_2$ are independent) as a function of $\theta$ for $n = 100$ in (a) and as a function of $n$ for $\theta = \pi/8$ in (b). We compute the mutual information with $k = 1$, $k = \sqrt{n}$ and $k = n/2$ for our test, and compare it to the HSIC in [10].

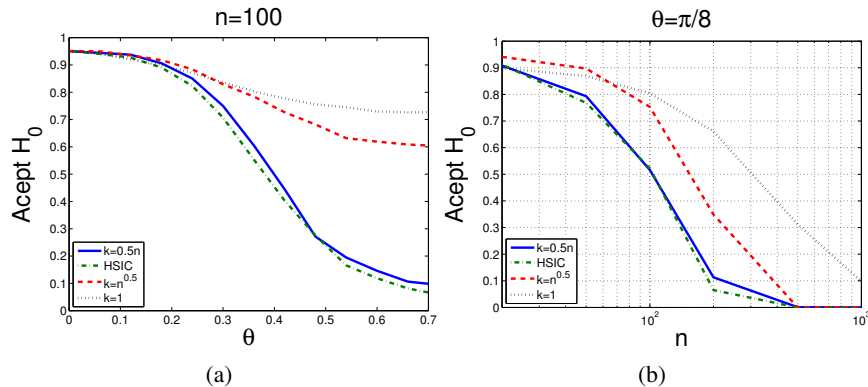

Figure 3: We plot the acceptance of the null hypothesis ($y_1$ and $y_2$ are independent) for a 95% confidence interval in (a) as a function of $\theta$ and in (b) as a function on (n). The solid line uses the mutual information estimate with $k = n/2$ and the dash-dotted line uses the HSIC. The dashed and dotted lines, respectively, use the mutual information estimate with $k = \sqrt{n}$ and $k = 1$.

The HSIC test and the mutual information estimate based test with $k = n/2$ perform equally well at predicting whether $y_1$ and $y_2$ are independent, while the test based on the mutual information estimates with $k = 1$ and $k = \sqrt{n}$ clearly underperforms. This example shows that if our goal is to predict whether two random variables are independent we are better off using HSIC or a nonconvergent estimate of the mutual information rather than trying to compute the mutual information as accurately as possible. Furthermore, in our test, the computational complexity of computing HSIC for $n = 5000$ is over 10 times more computationally costly (running time) than computing the mutual information for $k = n/2^2$.

As we saw in the case of the divergence estimate in Figure 1, mutual information is more accurately estimated when $k = 1$, but at the cost of a higher variance. If our objective is to estimate the mutual information (or the divergence), we should use a small value of $k$, ideally $k = 1$. However, if we are interested in assessing whether two random variables are independent, it is better to use $k = n/2$, because the variance of the estimate is much lower, even though it does not converge to the mutual information (or the divergence).

## 5  Conclusions

We have proved that the estimates of the differential entropy, mutual information and divergence based on $k$-nn density estimation for finite $k$ converge almost surely, even though the density estimate does not converge. The previous literature could only prove mean-squared consistency and it required imposing some constraints over the density models. The proof in this paper relies on describing the limiting distribution of $p(\mathbf{x})/\widehat{p}_k(\mathbf{x})$. This limiting distribution can be easily described using waiting-times distributions, such as the exponential or the Erlang distributions.

We have shown, experimentally, that fixing $k = 1$ achieves the fastest convergence rate, at the expense of a higher variance for our estimator. The divergence, mutual information and differential entropy estimates using $k = 1$ are much better than the estimates using $k = \sqrt{n}$, even though for $k = \sqrt{n}$ we can prove that $\widehat{p}_k(\mathbf{x})$ converges to $p(\mathbf{x})$ while for finite $k$ this convergences does not occur.

Finally, if we are interested in solving the two-sample problem or assessing if two random variables are independent, it is best to fix $k$ to a fraction of $n$ (we have used $k = n/2$ in our experiments), although in this case the estimates do not converge to the true value. Nevertheless, their variances are significantly lower, which allows our tests to perform better. The tests with $k = n/2$ perform as well as the MMD test for solving the two sample problem and the HSIC for assessing independence.

## Acknowledgment

Fernando Prez-Cruz is supported by Marie Curie Fellowship 040883-AI-COM. This work was partially funded by Spanish government (Ministerio de Educación y Ciencia TEC2006-13514-C02-01/TCM.

## Footnotes

[1]See [22] for a detailed discussion on mutual information estimation in neuroscience.

[2]For computing HSIC test we use A. Gretton code in http://www.kyb.mpg.de/bs/people/arthur/indep.htm and for finding the $k$-nn we use the *sort* function in Matlab.

## References

[1] N. Anderson, P. Hall, and D. Titterington. Two-sample test statistics for measuring discrepancies between two multivariate probability density functions using kernel-based density estimates. *Journal of Multivariate Analysis*, 50(1):41–54, 7 1994.

[2] F. R. Bach and M. I. Jordan. Kernel independent component analysis. *JMLR*, 3:1–48, 2004.

[3] K. Balakrishnan and A. P. Basu. *The Exponential Distribution: Theory, Methods and Applications*. Gordon and Breach Publishers, Amsterdam, Netherlands, 1996.

[4] J. Beirlant, E. Dudewicz, L. Gyorfi, and E. van der Meulen. Nonparametric entropy estimation: An overview. *nternational Journal of the Mathematical Statistics Sciences*, pages 17–39, 1997.

[5] T. M. Cover and J. A. Thomas. *Elements of Information Theory*. Wiley, New York, USA, 1991.

[6] G. A. Darbellay and I. Vajda. Estimation of the information by an adaptive partitioning of the observation space. *IEEE Trans. Information Theory*, 45(4):1315–1321, 5 1999.

[7] R. L. Dobrushin. A simplified method for experimental estimate of the entropy of a stationary sequence. *Theory of Probability and its Applications*, (4):428–430, 1958.

[8] F. Fleuret. Fast binary feature selection with conditional mutual information. *JMLR*, 5:1531–1555, 2004.

[9] A. Gretton, K. M. Borgwardt, M. Rasch, B. Schölkopf, and A. Smola. A kernel method for the two-sample-problem. In B. Schölkopf, J. Platt, and T. Hofmann, editors, *Advances in Neural Information Processing Systems 19*, Cambridge, MA, 2007. MIT Press.

[10] A. Gretton, K. Fukumizu, C. H. Teo, L. Song, B. Schölkopf, and A. Smola. A kernel statistical test of independence. In J.C. Platt, D. Koller, Y. Singer, and S. Roweis, editors, *Advances in Neural Information Processing Systems 20*, Cambridge, MA, 2008. MIT Press.

[11] G.R. Grimmett and D.R. Stirzaker. *Probability and Random Processes*. Oxford University Press, Oxford, UK, 3 edition, 2001.

[12] Julian Havil. *Gamma: Exploring Euler's Constant*. Princeton University Press, New York, USA, 2003.

[13] S. Mallela I. S. Dhillon and R. Kumar. A divisive information-theoretic feature clustering algorithm for text classification. *JMLR*, 3:1265–1287, 3 2003.

[14] Leonard Kleinrock. *Queueing Systems. Volume 1: Theory*. Wiley, New York, USA, 1975.

[15] L. F. Kozachenko and N. N. Leonenko. Sample estimate of the entropy of a random vector. *Problems Inform. Transmission*, 23(2):95–101, 4 1987.

[16] A. Kraskov, H. Stögbauer, and P. Grassberger. Estimating mutual information. *Physical Review E*, 69(6):1–16, 6 2004.

[17] S. Kullback and R. A. Leibler. On information and sufficiency. *Ann. Math. Stats.*, 22(1):79–86, 3 1951.

[18] N. N. Leonenko, L. Pronzato, and V. Savani. A class of renyi information estimators for multidimensional densities. *Annals of Statistics*, 2007. Submitted.

[19] P. J. Moreno, P. P. Ho, and N. Vasconcelos. A kullback-leibler divergence based kernel for svm classification in multimedia applications. Technical Report HPL-2004-4, HP Laboratories, 2004.

[20] X. Nguyen, M. J. Wainwright, and M. I. Jordan. Nonparametric estimation of the likelihood ratio and divergence functionals. In *IEEE Int. Symp. Information Theory*, Nice, France, 6 2007.

[21] X. Nguyen, M. J. Wainwright, and M. I. Jordan. Estimating divergence functionals and the likelihood ratio by penalized convex risk minimization. In J.C. Platt, D. Koller, Y. Singer, and S. Roweis, editors, *Advances in Neural Information Processing Systems 20*, Cambridge, MA, 2008. MIT Press.

[22] L. Paninski. Estimation of entropy and mutual information. *Neural Compt*, 15(6):1191–1253, 6 2003.

[23] C. E. Shannon. A mathematical theory of communication. *Bell System Tech. J.*, pages 379–423, 1948.

[24] K. Torkkola. Feature extraction by non parametric mutual information maximization. *JMLR*, 3:1415–1438, 2003.

[25] Q. Wang, S. Kulkarni, and S. Verdú. Divergence estimation of continuous distributions based on data-dependent partitions. *IEEE Trans. Information Theory*, 51(9):3064–3074, 9 2005.

[26] Q. Wang, S. Kulkarni, and S. Verdú. A nearest-neighbor approach to estimating divergence between continuous random vectors. In *IEEE Int. Symp. Information Theory*, Seattle, USA, 7 2006.

